# Dynamic Foreground/Background Extraction from Images and Videos using Random Patches

**Le Lu**[*]
Integrated Data Systems Department
Siemens Corporate Research
Princeton, NJ 08540
le-lu@siemens.com

**Gregory Hager**
Department of Computer Science
Johns Hopkins University
Baltimore, MD 21218
hager@cs.jhu.edu

## Abstract

In this paper, we propose a novel exemplar-based approach to extract dynamic foreground regions from a changing background within a collection of images or a video sequence. By using image segmentation as a pre-processing step, we convert this traditional pixel-wise labeling problem into a lower-dimensional supervised, binary labeling procedure on image segments. Our approach consists of three steps. First, a set of random image patches are spatially and adaptively sampled within each segment. Second, these sets of extracted samples are formed into two "bags of patches" to model the foreground/background appearance, respectively. We perform a novel bidirectional consistency check between new patches from incoming frames and current "bags of patches" to reject outliers, control model rigidity and make the model adaptive to new observations. Within each bag, image patches are further partitioned and resampled to create an evolving appearance model. Finally, the foreground/background decision over segments in an image is formulated using an aggregation function defined on the similarity measurements of sampled patches relative to the foreground and background models. The essence of the algorithm is conceptually simple and can be easily implemented within a few hundred lines of Matlab code. We evaluate and validate the proposed approach by extensive real examples of the object-level image mapping and tracking within a variety of challenging environments. We also show that it is straightforward to apply our problem formulation on non-rigid object tracking with difficult surveillance videos.

## 1   Introduction

In this paper, we study the problem of object-level figure/ground segmentation in images and video sequences. The core problem can be defined as follows: Given an image $\mathbb{X}$ with known figure/ground labels $\mathbb{L}$, infer the figure/ground labels $\mathbb{L}'$ of a new image $\mathbb{X}'$ closely related to $\mathbb{X}$. For example, we may want to extract a walking person in an image using the figure/ground mask of the same person in another image of the same sequence. Our approach is based on training a classifier from the appearance of a pixel and its surrounding context (i.e., an image patch centered at the pixel) to recognize other similar pixels across images. To apply this process to a video sequence, we also evolve the appearance model over time.

A key element of our approach is the use of a prior segmentation to reduce the complexity of the segmentation process. As argued in [22], image segments are a more natural primitive for image modeling than pixels. More specifically, an image segmentation provides a natural dimensional reduction from the spatial resolution of the image to a much smaller set of spatially compact and relatively homogeneous regions. We can then focus on representing the appearance characteristics

---

[*]The work has been done while the first author was a graduate student in Johns Hopkins University.

of these regions. Borrowing a term from [22], we can think of each region as a "superpixel" which represents a complex connected spatial region of the image using a rich set of derived image features. We can then consider how to classify each superpixel (i.e. image segment) as foreground or background, and then project this classification back into the original image to create the pixel-level foreground-background segmentation we are interested in.

The original superpixel representation in [22, 19, 18] is a feature vector created from the image segment's color histogram [19], filter bank responses [22], oriented energy [18] and contourness [18]. These features are effective for image segmentation [18], or finding perceptually important boundaries from segmentation by supervised training [22]. However, as shown in [17], those parameters do not work well for matching different classes of image regions from different images. Instead, we propose using a set of spatially randomly sampled image patches as a non-parametric, statistical superpixel representation. This non-parametric "bag of patches" model[1] can be easily and robustly evolved with the spatial-temporal appearance information from video, while maintaining the model size (the number of image patches per bag) using adaptive sampling. Foreground/background classification is then posed as the problem of matching sets of random patches from the image with these models. Our *major contributions* are demonstrating the effectiveness and computational simplicity of a nonparametric random patch representation for semantically labelling superpixels and a novel bidirectional consistency check and resampling strategy for robust foreground/background appearance adaptation over time.

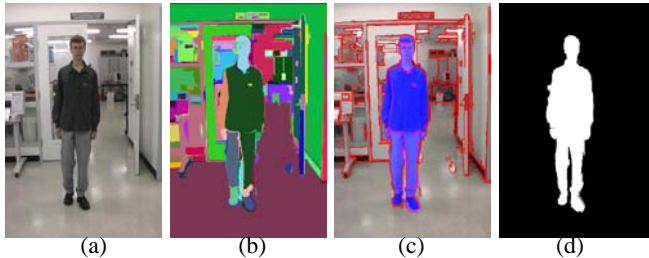

|     (a)     |     (b)     |     (c)     |     (d)     |

Figure 1: (a) An example indoor image, (b) the segmentation result using [6] coded in random colors, (c) the boundary pixels between segments shown in red, the image segments associated with the foreground, a walking person here, shown in blue, (d) the associated foreground/background mask. Notice that the color in (a) is not very saturated. This is a common fact in our indoor experiments without any specific lighting controls.

We organize the paper as follows. We first address several image patch based representations and the associated matching methods are described. In section 3, the algorithm used in our approach is presented with details. We demonstrate the validity of the proposed approach using experiments on real examples of the object-level figure/ground image mapping and non-rigid object tracking under dynamic conditions from videos of different resolutions in section 4. Finally, we summarize the contributions of the paper and discuss possible extensions and improvements.

## 2 Image Patch Representation and Matching

Building stable appearance representations of images patches is fundamental to our approach. There are many derived features that can be used to represent the appearance of an image patch. In this paper, we evaluate our algorithm based on: 1) an image patch's raw RGB intensity vector, 2) mean color vector, 3) color + texture descriptor (filter bank response or Haralick feature [17]), and 4) PCA, LDA and NDA (Nonparametric Discriminant Analysis) features [7, 3] on the raw RGB vectors. For completeness, we give a brief description of each of these techniques.

**Texture descriptors**: To compute texture descriptions, we first apply the *Leung-Malik (LM) filter bank* [13] which consists of 48 isotropic and anisotropic filters with 6 directions, 3 scales and 2 phases. Thus each image patch is represented by a 48 component feature vector. The *Haralick texture descriptor* [10] was used for image classification in [17]. Haralick features are derived from the Gray Level Co-occurrence Matrix, which is a tabulation of how often different combinations of pixel brightness values (grey levels) occur in an image region. We selected 5 out of 14 texture

descriptors [10] including dissimilarity, Angular Second Moment (ASM), mean, standard deviation (STD) and correlation. For details, refer to [10, 17].

**Dimension reduction representations**: The *Principal Component Analysis* (PCA) algorithm is used to reduce the dimensionality of the raw color intensity vectors of image patches. PCA makes no prior assumptions about the labels of data. However, recall that we construct the "bag of patches" appearance model from sets of labelled image patches. This supervised information can be used to project the bags of patches into a subspace where they are best separated using *Linear discriminant Analysis* (LDA) or *Nonparametric Discriminant Analysis* (NDA) algorithm [7, 3] by assuming Gaussian or Non-Gaussian class-specific distributions.

**Patch matching**: After image patches are represented using one of the above methods, we must match them against the foreground/background models. There are 2 methods investigated in this paper: the nearest neighbor matching using Euclidean distance and KDE (Kernel Density Estimation) [12] in PCA/NDA subspaces. For nearest-neighbor matching, we find, for each patch $p$, its nearest neighbors $p_n^F$, $p_n^B$ in foreground/background bags, and then compute $d_p^F = \parallel p - p_n^F \parallel$, $d_p^B = \parallel p - p_n^B \parallel$. On the other hand, an image patch's matching scores $m_p^F$ and $m_p^B$ are evaluated as probability density values from the KDE functions $KDE(p, \Omega^F)$ and $KDE(p, \Omega^B)$ where $\Omega^{F|B}$ are bags of patch models. Then the segmentation-level classification is performed as section 3.2.

# 3   Algorithms

We briefly summarize our labeling algorithm as follows. We assume that each image of interest has been segmented into spatial regions. A set of random image patches are spatially and adaptively sampled within each segment. These sets of extracted samples are formed into two "bags of patches" to model the foreground/background appearance respectively. The foreground/background decision for any segment in a new image is computed using one of two aggregation functions on the appearance similarities from its inside image patches to the foreground and background models. Finally, for videos, within each bag, new patches from new frames are integrated through a robust bidirectional consistency check process and all image patches are then partitioned and resampled to create an evolving appearance model. As described below, this process prune classification inaccuracies in the nonparametric image patch representations and adapts them towards current changes in foreground/background appearances for videos. We describe each of these steps for video tracking of foreground/background segments in more detail below, and for image matching, which we treat as a special case by simply omitting step 3 and 4 in Figure 2.

---

**Non-parametric Patch Appearance Modelling-Matching Algorithm**

*inputs:* Pre-segmented Images $\mathbb{X}_t, t = 1, 2, ..., T$; Label $\mathbb{L}_1$

*outputs:* Labels $\mathbb{L}_t, t = 2, ..., T$; 2 "bags of patches" appearance model for foreground/background $\Omega_T^{F|B}$

1.  Sample segmentation-adaptive random image patches $\{\mathcal{P}_1\}$ from image $\mathbb{X}_1$.
2.  Construct 2 new bags of patches $\Omega_1^{F|B}$ for foreground/background using patches $\{\mathcal{P}_1\}$ and label $\mathbb{L}_1$; set $t = 1$.
3.  $t = t + 1$; Sample segmentation-adaptive random image patches $\{\mathcal{P}_t\}$ from image $\mathbb{X}_t$; match $\{\mathcal{P}_t\}$ with $\Omega_{t-1}^{F|B}$ and classify segments of $\mathbb{X}_t$ to generate label $\mathbb{L}_t$ by aggregation.
4.  Classify and reject ambiguous patch samples, probable outliers and redundant appearance patch samples from new extracted image patches $\{\mathcal{P}_t\}$ against $\Omega_{t-1}^{F|B}$; Then integrate the filtered $\{\mathcal{P}_t\}$ into $\Omega_{t-1}^{F|B}$ and evaluate the probability of survival $p_s$ for each patch inside $\Omega_{t-1}^{F|B}$ against the original unprocessed $\{\mathcal{P}_t\}$ (Bidirectional Consistency Check).
5.  Perform the random partition and resampling process according to the normalized product of probability of survival $p_s$ and partition-wise sampling rate $\gamma'$ inside $\Omega_{t-1}^{F|B}$ to generate $\Omega_t^{F|B}$.
6.  If $t = T$, output $\mathbb{L}_t, t = 2, ..., T$ and $\Omega_T^{F|B}$; exit. If $t < T$, go to (3).

---

Figure 2: Non-parametric Patch Appearance Modelling-Matching Algorithm

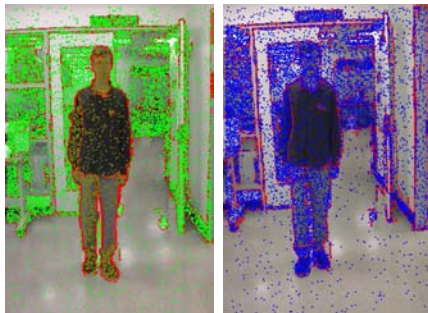

Figure 3: **Left:** Segment adaptive random patch sampling from an image with known figure/ground labels. Green dots are samples for background; dark brown dots are samples for foreground. **Right:** Segment adaptive random patch sampling from a new image for figure/ground classification, shown as blue dots.

## 3.1 Sample Random Image Patches

We first employ an image segmentation algorithm[2] [6] to pre-segment all the images or video frames in our experiments. A typical segmentation result is shown in Figure 1. We use $\mathbb{X}_t, t = 1, 2, ..., T$ to represent a sequence of video frames. Given an image segment, we formulate its representation as a distribution on the appearance variation over all possible extracted image patches inside the segment. To keep this representation to a manageable size, we approximate this distribution by sampling a random subset of patches.

We denote an image segment as $\mathcal{S}_i$ with $\mathcal{S}_i^F$ for a foreground segment, and $\mathcal{S}_i^B$ for a background segment, where $i$ is the index of the (foreground/background)image segment within an image. Accordingly, $\mathcal{P}_i$, $\mathcal{P}_i^F$ and $\mathcal{P}_i^B$ represent a set of random image patches sampled from $\mathcal{S}_i$, $\mathcal{S}_i^F$ and $\mathcal{S}_i^B$ respectively. The cardinality $\mathcal{N}_i$ of an image segment $\mathcal{S}_i$ generated by [6] typically ranges from 50 to thousands. However small or large superpixels are expected to have roughly the same amount of uniformity. Therefore the sampling rate $\gamma_i$ of $\mathcal{S}_i$, defined as $\gamma_i = size(\mathcal{P}_i)/\mathcal{N}_i$, should decrease with increasing $\mathcal{N}_i$. For simplicity, we keep $\gamma_i$ as a constant for all superpixels, unless $\mathcal{N}_i$ is above a predefined threshold $\tau$, (typically $2500 \sim 3000$), above which $size(\mathcal{P}_i)$ is held fixed. This sampling adaptivity is illustrated in Figure 3. Notice that large image segments have much more sparsely sampled patches than small image segments. From our experiments, this adaptive spatial sampling strategy is sufficient to represent image segments of different sizes.

## 3.2 Label Segments by Aggregating Over Random Patches

For an image segment $\mathcal{S}_i$ from a new frame to be classified, we again first sample a set of random patches $\mathcal{P}_i$ as its representative set of appearance samples. For each patch $p \in \mathcal{P}_i$, we calculate its distances $d_p^F, d_p^B$ or matching scores $m_p^B, m_p^F$ towards the foreground and background appearance models respectively as described in Section 2.

The decision of assigning $\mathcal{S}_i$ to foreground or background, is an aggregating process over all $\{d_p^F, d_p^B\}$ or $\{m_p^B; m_p^F\}$ where $p \in \mathcal{P}_i$. Since $\mathcal{P}_i$ is considered as a set of i.i.d. samples of the appearance distribution of $\mathcal{S}_i$, we use the average of $\{d_p^F, d_p^B\}$ or $\{m_p^B; m_p^F\}$ (ie. first-order statistics) as its distances $D_{\mathcal{P}_i}^F, D_{\mathcal{P}_i}^B$ or fitness values $M_{\mathcal{P}_i}^F, M_{\mathcal{P}_i}^B$ with the foreground/background model. In terms of distances $\{d_p^F, d_p^B\}$, $D_{\mathcal{P}_i}^F = mean_{p \in \mathcal{P}_i}(d_p^F)$ and $D_{\mathcal{P}_i}^B = mean_{p \in \mathcal{P}_i}(d_p^B)$. Then the segment's foreground/background fitness is set as the inverse of the distances: $M_{\mathcal{P}_i}^F = 1/D_{\mathcal{P}_i}^F$ and $M_{\mathcal{P}_i}^B = 1/D_{\mathcal{P}_i}^B$. In terms of KDE matching scores $\{m_p^B; m_p^F\}$, $M_{\mathcal{P}_i}^F = mean_{p \in \mathcal{P}_i}(m_p^F)$ and $M_{\mathcal{P}_i}^B = mean_{p \in \mathcal{P}_i}(m_p^B)$. Finally, $\mathcal{S}_i$ is classified as foreground if $M_{\mathcal{P}_i}^F > M_{\mathcal{P}_i}^B$, and vice versa. The *Median* robust operator can also be employed in our experiments, without noticeable difference in performance. Another choice is to classify each $p \in \mathcal{P}_i$ from $m_p^B$ and $m_p^F$, then vote the majority foreground/background decision for $\mathcal{S}_i$. The performance is similar with *mean* and *median*.

## 3.3 Construct a Robust Online Nonparametric Foreground/Background Appearance Model with Temporal Adaptation

From sets of random image patches extracted from superpixels with known figure/ground labels, 2 foreground/background "bags of patches" are composed. The bags are the non-parametric form of the foreground/background appearance distributions. When we intend to "track" the figure/ground model sequentially though a sequence, these models need to be updated by integrating new image patches extracted from new video frames. However the size (the number of patches) of the bag will be unacceptably large if we do not also remove the some redundant information over time. More importantly, imperfect segmentation results from [6] can cause inaccurate segmentation level figure/ground labels. For robust image patch level appearance modeling of $\Omega_t$, we propose a novel bidirectional consistency check and resampling strategy to tackle various noise and labelling uncertainties.

More precisely, we classify new extracted image patches $\{\mathcal{P}_t\}$ as $\{\mathcal{P}_t^F\}$ or $\{\mathcal{P}_t^B\}$ according to $\Omega_{t-1}^{F|B}$; and reject ambiguous patch samples whose distances $d_p^F, d_p^B$ towards respective $\Omega_{t-1}^{F|B}$ have no good contrast (simply, the ratio between $d_p^F$ and $d_p^B$ falls into the range of 0.8 to 1/0.8). We further sort the distance list of the newly classified foreground patches $\{\mathcal{P}_t^F\}$ to $\Omega_{t-1}^F$, filter out image patches on the top of the list which have too large distances and are probably to be outliers, and ones from the bottom of the list which have too small distances and contain probably redundant appearances compared with $\Omega_{t-1}^F$[3]. We perform the same process with $\{\mathcal{P}_t^B\}$ according to $\Omega_{t-1}^B$. Then the filtered $\{\mathcal{P}_t\}$ are integrated into $\Omega_{t-1}^{F|B}$ to form $\Omega_{t-1}^{F'|B'}$, and we evaluate the probability of survival $p_s$ for each patch inside $\Omega_{t-1}^{F'|B'}$ against the original unprocessed $\{\mathcal{P}_t\}$ with their labels[4].

Next, we cluster all image patches of $\Omega_{t-1}^{F'|B'}$ into $k$ partitions [8], and randomly resample image patches within each partition. This is roughly equivalent to finding the modes of an arbitrary distribution and sampling for each mode. Ideally, the resampling rate $\gamma'$ should decrease with increasing partition size, similar to the segment-wise sampling rate $\gamma$. For simplicity, we define $\gamma'$ as a constant value for all partitions, unless setting a threshold $\tau'$ to be the minimal required size[5] of partitions after resampling. If we perform resampling directly over patches without partitioning, some modes of the appearance distribution may be mistakenly removed. This strategy represents all partitions with sufficient number of image patches, regardless of their different sizes. In all, we resample image patches of $\Omega_{t-1}^{F|B}$, according to the normalized product of probability of survival $p_s$ and partition-wise sampling rate $\gamma'$, to generate $\Omega_t^{F|B}$. By approximately fixing the expected bag model size, the number of image patches extracted from a certain frame $\mathbb{X}_t$ in the bag decays exponentially in time.

The problem of partitioning image patches in the bag can be formulated as the NP-hard *k-center* problem. The definition of *k-center* is as follows: given a data set of $n$ points and a predefined cluster number $k$, find a partition of the points into $k$ subgroups $\mathcal{P}_1, \mathcal{P}_2, ..., \mathcal{P}_k$ and the data centers $c_1, c_2, ..., c_k$, to minimize the maximum radius of clusters $\max_i \max_{p \in \mathcal{P}_i} \| p - c_i \|$, where $i$ is the index of clusters. Gonzalez [8] proposed an efficient greedy algorithm, *farthest-point clustering*, which proved to give an approximation factor of 2 of the optimum. The algorithm operates as follows: pick a random point $p_1$ as the first cluster center and add it to the center set $C$; for iterations $i = 2, ..., k$, find the point $p_i$ with the farthest distance to the current center set $C$: $d_i(p_i, C) = \min_{c \in C} \| p_i - c \|$ and add $p_i$ to set $C$; finally assign data points to its nearest center and recompute the means of clusters in $C$. Compared with the popular k-means algorithm, this algorithm is computationally efficient and theoretically bounded[6]. In this paper, we employ the Eu-

clidean distance between an image patch and a cluster center, using the raw RGB intensity vector or the feature representations discussed in section 2.

## 4  Experiments

We have evaluated the image patch representations described in Section 2 for figure/ground mapping between pairs of image on video sequences taken with both static and moving cameras. Here we summarize our results.

### 4.1  Evaluation on Object-level Figure/Ground Image Mapping

We first evaluate our algorithm on object-level figure/ground mapping between pairs of images under eight configurations of different image patch representations and matching criteria. They are listed as follows: the nearest neighbor distance matching on the image patch's mean color vector (*MCV*); raw color intensity vector of regular patch scanning (*RCV*) or segment-adaptive patch sampling over image (*SCV*); color + filter bank response (*CFB*); color + Haralick texture descriptor (*CHA*); PCA feature vector (*PCA*); NDA feature vector (*NDA*) and kernel density evaluation on PCA features (*KDE*). In general, $8000 \sim 12000$ random patches are sampled per image. There is no apparent difference on classification accuracy for the patch size ranging from 9 to 15 pixels and the sample rate from 0.02 to 0.10. The PCA/NDA feature vector has 20 dimensions, and KDE is evaluated on the first $3 \sim 6$ PCA features.

Because the foreground figure has fewer of pixels than background, we conservatively measure the classification accuracy from the foreground's detection precision and recall on pixels. Precision is the ratio of the number of correctly detected foreground pixels to the total number of detected foreground pixels; recall is is the ratio of the number of correctly detected foreground pixels to the total number of foreground pixels in the image. The patch size is 11 by 11 pixels, and the segment-wise patch sampling rate $\gamma$ is fixed as 0.06, unless stated otherwise. Using 40 pairs of ($720 \times 480$) images with the labelled figure/ground segmentation, we compare their average classification accuracies in Tables 1.

| MCV | RCV | SCV | CFB | CHA | PCA | NDA | KDE |
|-----|-----|-----|-----|-----|-----|-----|-----|
| 0.46 | 0.81 | 0.97 | 0.92 | 0.89 | 0.93 | 0.96 | 0.69 |
| 0.28 | 0.89 | 0.95 | 0.85 | 0.81 | 0.85 | 0.87 | 0.98 |

Table 1: Evaluation on classification accuracy (ratio). The first row is precision; the second row is recall.

For figure/ground extraction accuracy, *SCV* has the best classification ratio using the raw color intensity vector without any dimension reduction. *MCV* has the worst accuracy, which shows that pixel-color leads to poor separability between figure and ground in our data set. Four feature based representations, *CFB, CHA, PCA, NDA* with reduced dimensions, have similar performance, whereas *NDA* is slightly better than the others. *KDE* tends to be more biased towards the foreground class because background usually has a wider, flatter density distribution. The superiority of *SCV* over *RCV* proves that our segment-wise random patch sampling strategy is more effective at classifying image segments than regularly scanning the image, even with more samples. As shown in Figure 4 (b), some small or irregularly-shaped image segments do not have enough patch samples to produce stable classifications.

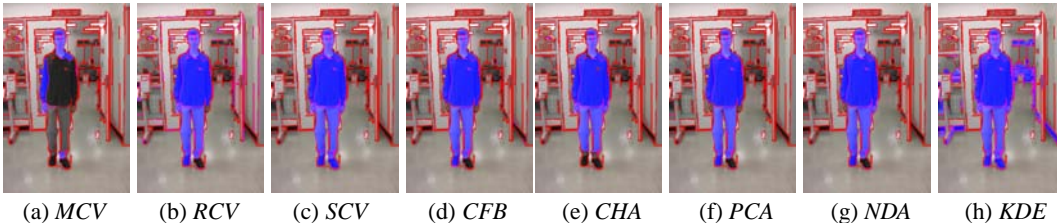

|     (a) *MCV*     (b) *RCV*     (c) *SCV*     (d) *CFB*     (e) *CHA*     (f) *PCA*     (g) *NDA*     (h) *KDE* |

Figure 4: An example of evaluation on object-level figure/ground image mapping. The labeled figure image segments are coded in blue.

### 4.2 Figure/Ground Segmentation Tracking with a Moving Camera

From Figure 4 (h), we see *KDE* tends to produce some false positives for the foreground. However the problem can be effectively tackled by multiplying the appearance KDE by the spatial prior which is also formulated as a KDE function of image patch coordinates. By considering videos with complex appearance-changing figure/ground, imperfect segmentation results [6] are not completely avoidable which can cause superpixel based figure/ground labelling errors. However our *robust bidirectional consistency check and resampling strategy*, as shown below, enables to successfully track the dynamic figure/ground segmentations in challenging scenarios with outlier rejection, model rigidity control and temporal adaptation (as described in section 3.3).

*Karsten.avi* shows a person walking in an uncontrolled indoor environment while tracked with a handheld camera. After we manually label the frame 1, the foreground/background appearance model starts to develop, classify new frames and get updated online. Eight Example tracking frames are shown in Figure 5. Notice that the significant non-rigid deformations and large scale changes of the walking person, while the original background is completely substituted after the subject turned his way. In frame 258, we manually eliminate some false positives of the figure. The reason for this failure is that some image regions which were behind the subject begin to appear when the person is walking from left to the center of image (starting from frame 220). Compared to the online foreground/background appearance models by then, these newly appearing image regions have quite different appearance from both the foreground and the background. Therefore the foreground's spatial prior dominates the classification. We leave this issue for future work.

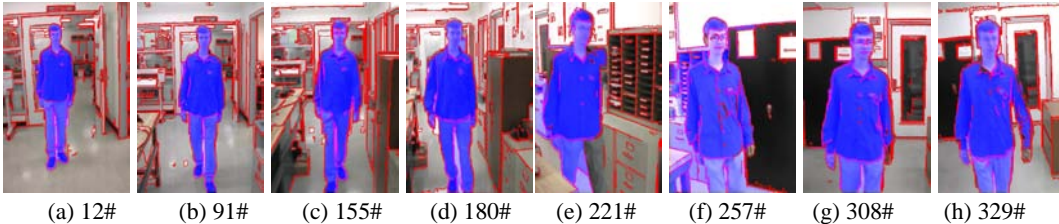

| (a) 12# | (b) 91# | (c) 155# | (d) 180# | (e) 221# | (f) 257# | (g) 308# | (h) 329# |

Figure 5: Eight example frames (720 by 480 pixels) from the video sequence *Karsten.avi* of 330 frames. The video is captured using a handheld Panasonic PV-GS120 in standard NTSC format. Notice that the significant non-rigid deformations and large scale changes of the walking person, while the original background is completely substituted after the subject turned his way. The red pixels are on the boundary of segments; the tracked image segments associated with the foreground walking person is coded in blue.

### 4.3 Non-rigid Object Tracking from Surveillance Videos

We can also apply our nonparametric treatment of dynamic random patches in Figure 2 into tracking non-rigid interested objects from surveillance videos. The difficult is that surveillance cameras normally capture small non-rigid figures, such as a walking person or running car, in low contrast and low resolution format. Thus to adapt our method to solve this problem, we make the following modifications. Because our task changes to localizing figure object automatically overtime, we can simply model figure/ground regions using rectangles and therefore no pre-segmentation [6] is needed. Random figure/ground patches are then extracted from the image regions within these two rectangles. Using two sets of random image patches, we train an online classifier for figure/ground classes at each time step, generate a figure appearance confidence map of classification for the next frame and, similarly to [1], apply mean shift [4] to find the next object location by mode seeking. In our problem solution, the temporal evolution of dynamic image patch appearance models are executed by the bidirectional consistency check and resampling described in section 3.3. Whereas [1] uses boosting for both temporal appearance model updating and classification, our online binary classification training can employ any off-the-shelf classifiers, such as k-Nearest Neighbors (KNN), support vector machine (SVM). Our results are favorably competitive to the state-of-the-art algorithms [1, 9], even under more challenging scenario.

### 5 Conclusion and Discussion

Although quite simple both conceptually and computationally, our algorithm of performing dynamic foreground-background extraction in images and videos using non-parametric appearance

models produces very promising and reliable results in a wide variety of circumstances. For tracking figure/ground segments, to our best knowledge, it is the first attempt to solve this difficult "video matting" problem [15, 25] by robust and automatic learning. For surveillance video tracking, our results are very competitive with the state-of-art [1, 9] under even more challenging conditions.

Our approach does not depend on an image segmentation algorithm that totally respects the boundaries of the foreground object. Our novel bidirectional consistency check and resampling process has been demonstrated to be effectively robust and adaptive. We leave the explorations on supervised dimension reduction and density modeling techniques on image patch sets, optimal random patch sampling strategy, and self-tuned optimal image patch size searching as our future work.

In this paper, we extract foreground/background by classifying on individual image segments. It might improve the figure/ground segmentation accuracy by modeling their spatial pairwise relationships as well. This problem can be further solved using generative or discriminative random field (MRF/DRF) model or the boosting method on logistic classifiers [11]. In this paper, we focus on learning binary dynamic appearance models by assuming figure/ground are somewhat distribution-wise separable. Other cues, as object shape regularization and motion dynamics for tracking, can be combined to improve performance.

## Footnotes

[1]Highly distinctive local features [16] are not the adequate substitutes for image patches. Their spatial sparseness nature limits their representativity within each individual image segment, especially for the nonrigid, nonstructural and flexible foreground/background appearance.

[2]Because we are not focused on image segmentation algorithms, we choose Felzenszwalb's segmentation code which generates good results and is publicly available at http://people.cs.uchicago.edu/~pff/segment/.

[3]Simply, we reject patches with distances $d_{p_t^F F}^F$ that are larger than $mean(d_{p_t^F F}^F) + \lambda * std(d_{p_t^F F}^F)$ or smaller than $mean(d_{p_t^F F}^F) - \lambda * std(d_{p_t^F F}^F)$ where $\lambda$ controls the range of accepting patch samples of $\Omega_{t-1}^{F|B}$, called *model rigidity*.

[4]For example, we compute the distance of each patch in $\Omega_{t-1}^{F'}$ to $\{\mathcal{P}_t^F\}$, and covert them as surviving probabilities using a exponential function over negative covariance normalized distances. Patches with smaller distances have higher survival chances during resampling; and vice versa. We perform the same process with $\Omega_{t-1}^{B'}$ according to $\{\mathcal{P}_t^B\}$.

[5]All image patches from partitions that are already smaller than $\tau'$ are kept during resampling.

[6]The random initialization of all $k$ centers and the local iterative smoothing process in k-means, which is time-consuming in high dimensional space and possibly converges to undesirable local minimum, are avoided.

# References

[1] S. Avidan, Ensemble Tracking, CVPR 2005.
[2] Y. Boykov and M. Jolly, Interactive Graph Cuts for Optimal boundary and Region Segmentation of Objects in n-d Images, ICCV, 2001.
[3] M. Bressan and J. Vitrià, Nonparametric discriminative analysis and nearest neighbor classification, Pattern Recognition Letter, 2003.
[4] D. Comaniciu and P. Meer, Mean shift: A robust approach toward feature space analysis. *IEEE Trans. PAMI*, 2002.
[5] A. Efros, T. Leung, Texture Synthesis by Non-parametric Sampling, ICCV, 1999.
[6] P. Felzenszwalb and D. Huttenlocher, Efficient Graph-Based Image Segmentation, *IJCV*, 2004.
[7] K. Fukunaga and J. Mantock, Nonparametric discriminative analysis, *IEEE Trans. on PAMI*, Nov. 1983.
[8] T. Gonzalez, Clustering to minimize the maximum intercluster distance, *Theoretical Computer Science*, 38:293-306, 1985.
[9] B. Han and L. Davis, On-Line Density-Based Appearance Modeling for Object Tracking, ICCV 2005.
[10] R. Haralick, K. Shanmugam, I. Dinstein, Texture features for image classification. *IEEE Trans. on SMC*, 1973.
[11] D. Hoiem, A. Efros and M. Hebert, Automatic Photo Pop-up, *SIGGRAPH*, 2005.
[12] A. Ihler, Kernel Density Estimation Matlab Toolbox, http://ssg.mit.edu/ ihler/code/kde.shtml.
[13] T. Leung and J. Malik, Representing and Recognizing the Visual Appearance of Materials using Three-Dimensional Textons, *IJCV*, 2001.
[14] Y. Li, J. Sun, C.-K. Tang and H.-Y. Shum, Lazy Snapping, *SIGGRAPH*, 2004.
[15] Y. Li, J. Sun and H.-Y. Shum. Video Object Cut and Paste, *SIGGRAPH*, 2005.
[16] D. Lowe, Distinctive image features from scale-invariant keypoints, *IJCV*, 2004.
[17] L. Lu, K. Toyama and G. Hager, A Two Level Approach for Scene Recognition, CVPR, 2005.
[18] J. Malik, S. Belongie, T. Leung, J. Shi, Contour and Texture Analysis for Image Segmentation, *IJCV*, 2001.
[19] D. Martin, C. Fowlkes, J. Malik, Learning to Detect Natural Image Boundaries Using Local Brightness, Color, and Texture Cues, *IEEE Trans. on PAMI*, 26(5):530-549, May 2004.
[20] A. Mittal and N. Paragios, Motion-based Background Substraction using Adaptive Kernel Density Estimation, CVPR, 2004.
[21] Eric Nowak, Frdric Jurie, Bill Triggs, Sampling Strategies for Bag-of-Features Image Classification, ECCV, 2006.
[22] X. Ren and J. Malik, Learning a classification model for segmentation, ICCV, 2003.
[23] C. Rother, V. Kolmogorov and A. Blake, Interactive Foreground Extraction using Iterated Graph Cuts, *SIGGRAPH*, 2004.
[24] Yaser Sheikh and Mubarak Shah, Bayesian Object Detection in Dynamic Scenes, CVPR, 2005.
[25] J. Wang, P. Bhat, A. Colburn, M. Agrawala and M. Cohen, Interactive Video Cutout. *SIGGRAPH*, 2005.
